# Efficient Learning using Forward-Backward Splitting

**John Duchi**
University of California Berkeley
jduchi@cs.berkeley.edu

**Yoram Singer**
Google
singer@google.com

## Abstract

We describe, analyze, and experiment with a new framework for empirical loss minimization with regularization. Our algorithmic framework alternates between two phases. On each iteration we first perform an *unconstrained* gradient descent step. We then cast and solve an instantaneous optimization problem that trades off minimization of a regularization term while keeping close proximity to the result of the first phase. This yields a simple yet effective algorithm for both batch penalized risk minimization and online learning. Furthermore, the two phase approach enables sparse solutions when used in conjunction with regularization functions that promote sparsity, such as $\ell_1$. We derive concrete and very simple algorithms for minimization of loss functions with $\ell_1$, $\ell_2$, $\ell_2^2$, and $\ell_\infty$ regularization. We also show how to construct efficient algorithms for mixed-norm $\ell_1/\ell_q$ regularization. We further extend the algorithms and give efficient implementations for very high-dimensional data with sparsity. We demonstrate the potential of the proposed framework in experiments with synthetic and natural datasets.

## 1 Introduction

Before we begin, we establish notation for this paper. We denote scalars by lower case letters and vectors by lower case bold letters, e.g. $\boldsymbol{w}$. The inner product of vectors $\boldsymbol{u}$ and $\boldsymbol{v}$ is denoted $\langle \boldsymbol{u}, \boldsymbol{v} \rangle$. We use $\|\boldsymbol{x}\|_p$ to denote the $p$-norm of the vector $\boldsymbol{x}$ and $\|\boldsymbol{x}\|$ as a shorthand for $\|\boldsymbol{x}\|_2$.

The focus of this paper is an algorithmic framework for regularized convex programming to minimize the following sum of two functions:

$$f(\boldsymbol{w}) + r(\boldsymbol{w}) \ , \tag{1}$$

where both $f$ and $r$ are convex bounded below functions (so without loss of generality we assume they are into $\mathbb{R}_+$). Often, the function $f$ is an empirical loss and takes the form $\sum_{i \in S} \ell_i(\boldsymbol{w})$ for a sequence of loss functions $\ell_i : \mathbb{R}^n \to \mathbb{R}_+$, and $r(\boldsymbol{w})$ is a regularization term that penalizes for excessively complex vectors, for instance $r(\boldsymbol{w}) = \lambda \|\boldsymbol{w}\|_p$. This task is prevalent in machine learning, in which a learning problem for decision and prediction problems is cast as a convex optimization problem. To that end, we propose a general and intuitive algorithm to minimize Eq. (1), focusing especially on derivations for and the use of non-differentiable regularization functions.

Many methods have been proposed to minimize general convex functions such as that in Eq. (1). One of the most general is the subgradient method [1], which is elegant and very simple. Let $\partial f(\boldsymbol{w})$ denote the subgradient set of $f$ at $\boldsymbol{w}$, namely, $\partial f(\boldsymbol{w}) = \{\boldsymbol{g} \mid \forall \boldsymbol{v} : f(\boldsymbol{v}) \geq f(\boldsymbol{w}) + \langle \boldsymbol{g}, \boldsymbol{v} - \boldsymbol{w} \rangle\}$. Subgradient procedures then minimize the function $f(\boldsymbol{w})$ by iteratively updating the parameter vector $\boldsymbol{w}$ according to the update rule $\boldsymbol{w}_{t+1} = \boldsymbol{w}_t - \eta_t \boldsymbol{g}_t^f$, where $\eta_t$ is a constant or diminishing step size and $\boldsymbol{g}_t^f \in \partial f(\boldsymbol{w}_t)$ is an arbitrary vector from the subgradient set of $f$ evaluated at $\boldsymbol{w}_t$. A slightly more general method than the above is the projected gradient method, which iterates

$$\boldsymbol{w}_{t+1} = \Pi_\Omega \left( \boldsymbol{w}_t - \eta_t \boldsymbol{g}_t^f \right) = \operatorname*{argmin}_{\boldsymbol{w} \in \Omega} \left\{ \left\| \boldsymbol{w} - (\boldsymbol{w}_t - \eta_t \boldsymbol{g}_t^f) \right\|_2^2 \right\}$$

where $\Pi_\Omega(\boldsymbol{w})$ is the Euclidean projection of $\boldsymbol{w}$ onto the set $\Omega$. Standard results [1] show that the (projected) subgradient method converges at a rate of $O(1/\varepsilon^2)$, or equivalently that the error $f(\boldsymbol{w}) - f(\boldsymbol{w}^\star) = O(1/\sqrt{T})$, given some simple assumptions on the boundedness of the subdifferential set and $\Omega$ (we have omitted constants dependent on $\|\partial f\|$ or $\dim(\Omega)$). Using the subgradient method to minimize Eq. (1) gives simple iterates of the form $\boldsymbol{w}_{t+1} = \boldsymbol{w}_t - \eta_t \boldsymbol{g}_t^f - \eta_t \boldsymbol{g}_t^r$, where $\boldsymbol{g}_t^r \in \partial r(\boldsymbol{w}_t)$. A common problem in subgradient methods is that if $r$ or $f$ is non-differentiable, the iterates of the subgradient method are very rarely at the points of non-differentiability. In the case of regularization functions such as $r(\boldsymbol{w}) = \|\boldsymbol{w}\|_1$, however, these points (zeros in the case of the $\ell_1$-norm) are often the true minima of the function. Furthermore, with $\ell_1$ and similar penalties, zeros are desirable solutions as they tend to convey information about the structure of the problem being solved [2, 3].

There has been a significant amount of work related to minimizing Eq. (1), especially when the function $r$ is a sparsity-promoting regularizer. We can hardly do justice to the body of prior work, and we provide a few references here to the research we believe is most directly related. The approach we pursue below is known as "forward-backward splitting" or a composite gradient method in the optimization literature and has been independently suggested by [4] in the context of sparse signal reconstruction, where $f(\boldsymbol{w}) = \|\boldsymbol{y} - A\boldsymbol{w}\|^2$, though they note that the method can apply to general convex $f$. [5] give proofs of convergence for forward-backward splitting in Hilbert spaces, though without establishing strong rates of convergence. The motivation of their paper is signal reconstruction as well. Similar projected-gradient methods, when the regularization function $r$ is no longer part of the objective function but rather cast as a constraint so that $r(\boldsymbol{w}) \leq \lambda$, are also well known [1]. [6] give a general and efficient projected gradient method for $\ell_1$-constrained problems. There is also a body of literature on regret analysis for online learning and online convex programming with convex constraints upon which we build [7, 8]. Learning sparse models generally is of great interest in the statistics literature, specifically in the context of consistency and recovery of sparsity patterns through $\ell_1$ or mixed-norm regularization across multiple tasks [2, 3, 9].

In this paper, we describe a general gradient-based framework, which we call FOBOS, and analyze it in batch and online learning settings. The paper is organized as follows. In the next section, we begin by introducing and formally defining the method, giving some simple preliminary analysis. We follow the introduction by giving in Sec. 3 rates of convergence for batch (offline) optimization. We then provide bounds for online convex programming and give a convergence rate for stochastic gradient descent. To demonstrate the simplicity and usefulness of the framework, we derive in Sec. 4 algorithms for several different choices of the regularizing function $r$. We extend these methods to be efficient in very high dimensional settings where the input data is sparse in Sec. 5. Finally, we conclude in Sec. 6 with experiments examining various aspects of the proposed framework, in particular the runtime and sparsity selection performance of the derived algorithms.

## 2 Forward-Looking Subgradients and Forward-Backward Splitting

In this section we introduce our algorithm, laying the framework for its strategy for online or batch convex programming. We originally named the algorithm Folos as an abbreviation for FOrward-LOoking Subgradient. Our algorithm is a distillation of known approaches for convex programming, in particular the Forward-Backward Splitting method. In order not to confuse readers of the early draft, we attempt to stay close to the original name and use the acronym FOBOS rather than Fobas. FOBOS is motivated by the desire to have the iterates $\boldsymbol{w}_t$ attain points of non-differentiability of the function $r$. The method alleviates the problems of non-differentiability in cases such as $\ell_1$-regularization by taking analytical minimization steps interleaved with subgradient steps. Put informally, FOBOS is analogous to the *projected* subgradient method, but replaces or augments the projection step with an instantaneous minimization problem for which it is possible to derive a closed form solution. FOBOS is succinct as each iteration consists of the following two steps:

$$\boldsymbol{w}_{t+\frac{1}{2}} = \boldsymbol{w}_t - \eta_t \boldsymbol{g}_t^f \tag{2}$$

$$\boldsymbol{w}_{t+1} = \operatorname*{argmin}_{\boldsymbol{w}} \left\{ \frac{1}{2} \left\| \boldsymbol{w} - \boldsymbol{w}_{t+\frac{1}{2}} \right\|^2 + \eta_{t+\frac{1}{2}} \, r(\boldsymbol{w}) \right\} \ . \tag{3}$$

In the above, $\boldsymbol{g}_t^f$ is a vector in $\partial f(\boldsymbol{w}_t)$ and $\eta_t$ is the step size at time step $t$ of the algorithm. The actual value of $\eta_t$ depends on the specific setting and analysis. The first step thus simply amounts to an unconstrained subgradient step with respect to the function $f$. In the second step we find a

new vector that interpolates between two goals: (i) stay close to the interim vector $\boldsymbol{w}_{t+\frac{1}{2}}$, and (ii) attain a low complexity value as expressed by $r$. Note that the regularization function is scaled by an interim step size, denoted $\eta_{t+\frac{1}{2}}$. The analyses we describe in the sequel determine the specific value of $\eta_{t+\frac{1}{2}}$, which is either $\eta_t$ or $\eta_{t+1}$. A key property of the solution of Eq. (3) is the necessary condition for optimality and gives the reason behind the name FOBOS. Namely, the zero vector must belong to subgradient set of the objective at the optimum $\boldsymbol{w}_{t+1}$, that is,

$$\boldsymbol{0} \in \partial \left\{ \frac{1}{2} \left\| \boldsymbol{w} - \boldsymbol{w}_{t+\frac{1}{2}} \right\|^2 + \eta_{t+\frac{1}{2}} \, r(\boldsymbol{w}) \right\} \bigg|_{\boldsymbol{w}=\boldsymbol{w}_{t+1}} .$$

Since $\boldsymbol{w}_{t+\frac{1}{2}} = \boldsymbol{w}_t - \eta_t \boldsymbol{g}_t^f$, the above property amounts to $\boldsymbol{0} \in \boldsymbol{w}_{t+1} - \boldsymbol{w}_t + \eta_t \boldsymbol{g}_t^f + \eta_{t+\frac{1}{2}} \partial r(\boldsymbol{w}_{t+1})$. This property implies that so long as we choose $\boldsymbol{w}_{t+1}$ to be the minimizer of Eq. (3), we are guaranteed to obtain a vector $\boldsymbol{g}_{t+1}^r \in \partial r(\boldsymbol{w}_{t+1})$ such that $\boldsymbol{0} = \boldsymbol{w}_{t+1} - \boldsymbol{w}_t + \eta_t \boldsymbol{g}_t^f + \eta_{t+\frac{1}{2}} \boldsymbol{g}_{t+1}^r$. We can understand this as an update scheme where the new weight vector $\boldsymbol{w}_{t+1}$ is a linear combination of the previous weight vector $\boldsymbol{w}_t$, a vector from the subgradient set of $f$ at $\boldsymbol{w}_t$, and a vector from the subgradient of $r$ evaluated at the yet to be determined $\boldsymbol{w}_{t+1}$. To recap, we can write $\boldsymbol{w}_{t+1}$ as

$$\boldsymbol{w}_{t+1} = \boldsymbol{w}_t - \eta_t \, \boldsymbol{g}_t^f - \eta_{t+\frac{1}{2}} \, \boldsymbol{g}_{t+1}^r, \tag{4}$$

where $\boldsymbol{g}_t^f \in \partial f(\boldsymbol{w}_t)$ and $\boldsymbol{g}_{t+1}^r \in \partial r(\boldsymbol{w}_{t+1})$. Solving Eq. (3) with $r$ above has two main benefits. First, from an algorithmic standpoint, it enables sparse solutions at virtually no additional computational cost. Second, the forward-looking gradient allows us to build on existing analyses and show that the resulting framework enjoys the formal convergence properties of many existing gradient-based and online convex programming algorithms.

## 3 Convergence and Regret Analysis of FOBOS

In this section we build on known results while using the forward-looking property of FOBOS to provide convergence rate and regret analysis. To derive convergence rates we set $\eta_{t+\frac{1}{2}}$ properly. As we show in the sequel, it is sufficient to set $\eta_{t+\frac{1}{2}}$ to $\eta_t$ or $\eta_{t+1}$, depending on whether we are doing online or batch optimization, in order to obtain convergence and low regret bounds. We provide proofs of all theorems in this paper, as well as a few useful technical lemmas, in the appendices, as the main foci of the paper are the simplicity of the method and derived algorithms and their experimental usefulness. The overall proof techniques all rely on the forward-looking property in Eq. (4) and moderately straightforward arguments with convexity and subgradient calculus.

Throughout the section we denote by $\boldsymbol{w}^\star$ the minimizer of $f(\boldsymbol{w}) + r(\boldsymbol{w})$. The first bounds we present rely only on the assumption that $\|\boldsymbol{w}^\star\| \le D$, though they are not as tight as those in the sequel. In what follows, define $\|\partial f(\boldsymbol{w})\| \triangleq \sup_{\boldsymbol{g} \in \partial f(\boldsymbol{w})} \|\boldsymbol{g}\|$. We begin by deriving convergence results under the fairly general assumption [10, 11] that the subgradients are bounded as follows:

$$\|\partial f(\boldsymbol{w})\|^2 \le A f(\boldsymbol{w}) + G^2, \quad \|\partial r(\boldsymbol{w})\|^2 \le A r(\boldsymbol{w}) + G^2 . \tag{5}$$

For example, any Lipschitz loss (such as the logistic or hinge/SVM) satisfies the above with $A = 0$ and $G$ equal to the Lipschitz constant; least squares satisfies Eq. (5) with $G = 0$ and $A = 4$.

**Theorem 1.** *Assume the following hold: (i) the norm of any subgradient from $\partial f$ and the norm of any subgradient from $\partial r$ are bounded as in Eq. (5), (ii) the norm of $\boldsymbol{w}^\star$ is less than or equal to $D$, (iii) $r(\boldsymbol{0}) = 0$, and (iv) $\frac{1}{2}\eta_t \le \eta_{t+1} \le \eta_t$. Then for a constant $c \le 4$ with $\boldsymbol{w}_1 = \boldsymbol{0}$ and $\eta_{t+\frac{1}{2}} = \eta_{t+1}$,*

$$\sum_{t=1}^{T} \left[ \eta_t \left( (1 - cA\eta_t) f(\boldsymbol{w}_t) - f(\boldsymbol{w}^\star) \right) + \eta_t \left( (1 - cA\eta_t) r(\boldsymbol{w}_t) - r(\boldsymbol{w}^\star) \right) \right] \le D^2 + 7G^2 \sum_{t=1}^{T} \eta_t^2 .$$

The proof of the theorem is in Appendix A. We also provide in the appendix a few useful corollaries. We provide one corollary below as it underscores that the rate of convergence $\approx \sqrt{T}$.

**Corollary 2** (Fixed step rate). *Assume that the conditions of Thm. 1 hold and that we run FOBOS for a predefined $T$ iterations with $\eta_t = \frac{D}{\sqrt{7T}G}$ and that $(1 - cA\frac{D}{\sqrt{7T}G}) > 0$. Then*

$$\min_{t \in \{1, \dots, T\}} f(\boldsymbol{w}_t) + r(\boldsymbol{w}_t) \le \frac{1}{T} \sum_{t=1}^{T} f(\boldsymbol{w}_t) + r(\boldsymbol{w}_t) \le \frac{3DG}{\sqrt{T} \left( 1 - \frac{cAD}{G\sqrt{7T}} \right)} + \frac{f(\boldsymbol{w}^\star) + r(\boldsymbol{w}^\star)}{1 - \frac{cAD}{G\sqrt{7T}}}$$

Bounds of the form we present above, where the point minimizing $f(\boldsymbol{w}_t) + r(\boldsymbol{w}_t)$ converges rather than the last point $\boldsymbol{w}_T$, are standard in subgradient optimization. This occurs since there is no way to guarantee a descent direction when using arbitrary subgradients (see, e.g., [12, Theorem 3.2.2]).

We next derive regret bounds for FOBOS in online settings in which we are given a sequence of functions $f_t : \mathbb{R}^n \to \mathbb{R}$. The goal is for the sequence of predictions $\boldsymbol{w}_t$ to attain low regret when compared to a single optimal predictor $\boldsymbol{w}^\star$. Formally, let $f_t(\boldsymbol{w})$ denote the loss suffered on the $t^{th}$ input loss function when using a predictor $\boldsymbol{w}$. The regret of an online algorithm which uses $\boldsymbol{w}_1, \ldots, \boldsymbol{w}_t, \ldots$ as its predictors w.r.t a fixed predictor $\boldsymbol{w}^\star$ while using a regularization function $r$ is

$$R_{f+r}(T) = \sum_{t=1}^{T} \left[ f_t(\boldsymbol{w}_t) + r(\boldsymbol{w}_t) - (f_t(\boldsymbol{w}^\star) + r(\boldsymbol{w}^\star)) \right] \ .$$

Ideally, we would like to achieve 0 regret to a stationary $\boldsymbol{w}^\star$ for arbitrary length sequences.

To achieve an online bound for a sequence of convex functions $f_t$, we modify arguments of [7]. We begin with a slightly different assignment for $\eta_{t+\frac{1}{2}}$: specifically, we set $\eta_{t+\frac{1}{2}} = \eta_t$. We have the following theorem, whose proof we provide in Appendix B.

**Theorem 3.** *Assume that $\|\boldsymbol{w}_t - \boldsymbol{w}^\star\| \leq D$ for all iterations and the norm of the subgradient sets $\partial f_t$ and $\partial r$ are bounded above by $G$. Let $c > 0$ an arbitrary scalar. Then the regret bound of FOBOS with $\eta_t = c/\sqrt{t}$ satisfies $R_{f+r}(T) \leq GD + \left( \frac{D^2}{2c} + 7G^2 c \right) \sqrt{T}$.*

For slightly technical reasons, the assumption on the boundedness of $\boldsymbol{w}_t$ and the subgradients is not actually restrictive (see Appendix A for details). It is possible to obtain an $O(\log T)$ regret bound for FOBOS when the sequence of loss functions $f_t(\cdot)$ or the function $r(\cdot)$ is strongly convex, similar to [8], by using the curvature of $f_t$ or $r$. While we can extend these results to FOBOS, we omit the extension for lack of space (though we do perform some experiments with such functions). Using the regret analysis for online learning, we can also give convergence rates for stochastic FOBOS, which are $O(\sqrt{T})$. Further details are given in Appendix B and the long version of this paper [13].

## 4 Derived Algorithms

We now give a few variants of FOBOS by considering different regularization functions. The emphasis of the section is on non-differentiable regularization functions that lead to sparse solutions. We also give simple extensions to apply FOBOS to mixed-norm regularization [9] that build on the first part of this section. For lack of space, we mostly give the resulting updates, skipping technical derivations. We would like to note that some of the following results were tacitly given in [4]. First, we make a few changes to notation. To simplify our derivations, we denote by $\boldsymbol{v}$ the vector $\boldsymbol{w}_{t+\frac{1}{2}} = \boldsymbol{w}_t - \eta_t \boldsymbol{g}_t^f$ and let $\tilde{\lambda}$ denote $\eta_{t+\frac{1}{2}} \cdot \lambda$. Using this notation the problem given in Eq. (3) can be rewritten as $\min_{\boldsymbol{w}} \frac{1}{2}\|\boldsymbol{w} - \boldsymbol{v}\|^2 + \tilde{\lambda}\, r(\boldsymbol{w})$. Lastly, we let $[z]_+$ denote $\max \{0, z\}$.

**FOBOS with $\ell_1$ regularization:** The update obtained by choosing $r(\boldsymbol{w}) = \lambda \|\boldsymbol{w}\|_1$ is simple and intuitive. The objective is decomposable into a sum of 1-dimensional convex problems of the form $\min_w \frac{1}{2}(w - v)^2 + \tilde{\lambda}|w|$. As a result, the components of the optimal solution $\boldsymbol{w}^\star = \boldsymbol{w}_{t+1}$ are computed from $\boldsymbol{w}_{t+\frac{1}{2}}$ as

$$w_{t+1,j} = \operatorname{sign}\left(w_{t+\frac{1}{2},j}\right)\left[\left|w_{t+\frac{1}{2},j}\right| - \tilde{\lambda}\right]_+ = \operatorname{sign}\left(w_{t,j} - \eta_t g_{t,j}^f\right)\left[\left|w_{t,j} - \eta_t g_{t,j}^f\right| - \lambda\eta_{t+\frac{1}{2}}\right]_+ \quad (6)$$

Note that this update leads to sparse solutions: whenever the absolute value of a component of $\boldsymbol{w}_{t+\frac{1}{2}}$ is smaller than $\tilde{\lambda}$, the corresponding component in $\boldsymbol{w}_{t+1}$ is set to zero. Eq. (6) gives a simple online and offline method for minimizing a convex $f$ with $\ell_1$ regularization. [10] recently proposed and analyzed the same update, terming it the "truncated gradient," though the analysis presented here stems from a more general framework. This update can also be implemented very efficiently when the support of $\boldsymbol{g}_t^f$ is small [10], but we defer details to Sec. 5, where we describe a unified view that facilitates an efficient implementation for all the regularization functions discussed in this paper.

**FOBOS with $\ell_2^2$ regularization:** When $r(\boldsymbol{w}) = \frac{\lambda}{2}\|\boldsymbol{w}\|_2^2$, we obtain a very simple optimization problem, $\min_{\boldsymbol{w}} \frac{1}{2}\|\boldsymbol{w} - \boldsymbol{v}\|^2 + \frac{1}{2}\tilde{\lambda}\|\boldsymbol{w}\|^2$. Differentiating the objective and setting the result equal to

zero, we have $\boldsymbol{w}^\star - \boldsymbol{v} + \tilde{\lambda}\boldsymbol{w}^\star = 0$, which, using the original notation, yields the update

$$\boldsymbol{w}_{t+1} = \frac{\boldsymbol{w}_t - \eta_t \boldsymbol{g}_t^f}{1 + \tilde{\lambda}} \quad . \tag{7}$$

Informally, the update simply shrinks $\boldsymbol{w}_{t+1}$ back toward the origin after each gradient-descent step.

FOBOS **with $\ell_2$ regularization:** A lesser used regularization function is the $\ell_2$ norm of the weight vector. By setting $r(\boldsymbol{w}) = \tilde{\lambda}\|\boldsymbol{w}\|$ we obtain the following problem: $\min_{\boldsymbol{w}} \frac{1}{2}\|\boldsymbol{w} - \boldsymbol{v}\|^2 + \tilde{\lambda}\|\boldsymbol{w}\|$. The solution of the above problem must be in the direction of $\boldsymbol{v}$ and takes the form $\boldsymbol{w}^\star = s\boldsymbol{v}$ where $s \geq 0$. The resulting second step of the FOBOS update with $\ell_2$ regularization amounts to

$$\boldsymbol{w}_{t+1} = \left[1 - \frac{\tilde{\lambda}}{\|\boldsymbol{w}_{t+\frac{1}{2}}\|}\right]_+ = \left[1 - \frac{\tilde{\lambda}}{\|\boldsymbol{w}_t - \eta_t \boldsymbol{g}_t^f\|}\right]_+ (\boldsymbol{w}_t - \eta_t \boldsymbol{g}_t^f) \quad .$$

$\ell_2$-regularization results in a zero weight vector under the condition that $\|\boldsymbol{w}_t - \eta_t \boldsymbol{g}_t^f\| \leq \tilde{\lambda}$. This condition is rather more stringent for sparsity than the condition for $\ell_1$, so it is unlikely to hold in high dimensions. However, it does constitute a very important building block when using a mixed $\ell_1/\ell_2$-norm as the regularization, as we show in the sequel.

FOBOS **with $\ell_\infty$ regularization:** We now turn to a less explored regularization function, the $\ell_\infty$ norm of $\boldsymbol{w}$. Our interest stems from the recognition that there are settings in which it is desirable to consider blocks of variables as a group (see below). We wish to obtain an efficient solution to

$$\min_{\boldsymbol{w}} \frac{1}{2}\|\boldsymbol{w} - \boldsymbol{v}\|^2 + \tilde{\lambda}\|\boldsymbol{w}\|_\infty \quad . \tag{8}$$

A solution to the dual form of Eq. (8) is well established. Recalling that the conjugate of the quadratic function is a quadratic function and the conjugate of the $\ell_\infty$ norm is the $\ell_1$ barrier function, we immediately obtain that the dual of the problem in Eq. (8) is $\max_{\boldsymbol{\alpha}} -\frac{1}{2}\|\boldsymbol{\alpha} - \boldsymbol{v}\|_2^2$ s.t. $\|\boldsymbol{\alpha}\|_1 \leq \tilde{\lambda}$. Moreover, the vector of dual variables $\boldsymbol{\alpha}$ satisfies the relation $\boldsymbol{\alpha} = \boldsymbol{v} - \boldsymbol{w}$. [6] describes a linear time algorithm for finding the optimal $\boldsymbol{\alpha}$ to this $\ell_1$-constrained projection, and the analysis there shows the optimal solution to Eq. (8) is $w_{t+1,j} = \mathrm{sign}(w_{t+\frac{1}{2},j}) \min\{|w_{t+\frac{1}{2},j}|, \theta\}$. The optimal solution satisfies $\theta = 0$ iff $\|\boldsymbol{w}_{t+\frac{1}{2}}\|_1 \leq \tilde{\lambda}$, and otherwise $\theta > 0$ and can be found in $O(n)$ steps.

**Mixed norms:** We saw above that when using either the $\ell_2$ or the $\ell_\infty$ norm as the regularizer we obtain an all zeros vector if $\|\boldsymbol{w}_{t+\frac{1}{2}}\|_2 \leq \tilde{\lambda}$ or $\|\boldsymbol{w}_{t+\frac{1}{2}}\|_1 \leq \tilde{\lambda}$, respectively. This phenomenon can be useful. For example, in multiclass categorization problems each class $s$ may be associated with a different weight vector $\boldsymbol{w}^s$. The prediction for an instance $\boldsymbol{x}$ is a vector $\langle \boldsymbol{w}^1, \boldsymbol{x} \rangle, \ldots, \langle \boldsymbol{w}^k, \boldsymbol{x} \rangle$, where $k$ is the number of classes, and the predicted class is $\mathrm{argmax}_j \langle \boldsymbol{w}^j, \boldsymbol{x} \rangle$. Since all the weight vectors operate over the same instance space, it may be beneficial to tie the weights corresponding to the same input feature: we would to zero the row of weights $w_j^1, \ldots, w_j^k$ *simultaneously*.

Formally, let $W$ represent an $n \times k$ matrix where the $j^{th}$ column of the matrix is the weight vector $\boldsymbol{w}^j$ associated with class $j$. Then the $i^{th}$ row contains weight of the $i^{th}$ feature for each class. The mixed $\ell_r/\ell_s$-norm [9] of $W$ is obtained by computing the $\ell_s$-norm of each row of $W$ and then applying the $\ell_r$-norm to the resulting $n$ dimensional vector, for instance, $\|W\|_{\ell_1/\ell_\infty} = \sum_{j=1}^n \max_j |W_{i,j}|$. In a mixed-norm regularized optimization problem, we seek the minimizer of $f(W) + \lambda\|W\|_{\ell_r/\ell_s}$. Given the specific variants of norms described above, the FOBOS update for the $\ell_1/\ell_\infty$ and the $\ell_1/\ell_2$ mixed-norms is readily available. Let $\bar{\boldsymbol{w}}^s$ be the $s^{th}$ row of $W$. Analogously to standard norm-based regularization, we use the shorthand $V = W_{t+\frac{1}{2}}$. For the $\ell_1/\ell_p$ mixed-norm, we need to solve

$$\min_W \frac{1}{2}\|W - V\|_{\mathrm{Fr}}^2 + \tilde{\lambda}\|W\|_{\ell_1/\ell_p} \equiv \min_{\bar{\boldsymbol{w}}^1, \ldots, \bar{\boldsymbol{w}}^k} \sum_{i=1}^n \left(\frac{1}{2}\|\bar{\boldsymbol{w}}^i - \bar{\boldsymbol{v}}^i\|_2^2 + \tilde{\lambda}\|\bar{\boldsymbol{w}}^i\|_p\right) \tag{9}$$

where $\bar{\boldsymbol{v}}^i$ is the $i^{th}$ row of $V$. It is immediate to see that the problem given in Eq. (9) is decomposable into $n$ separate problems of dimension $k$, each of which can be solved by the procedures described in the prequel. The end result of solving these types of mixed-norm problems is a sparse matrix with numerous zero rows. We demonstrate the merits of FOBOS with mixed-norms in Sec. 6.

# 5 Efficient implementation in high dimensions

In many settings, especially online learning, the weight vector $\boldsymbol{w}_t$ and the gradients $\boldsymbol{g}_t^f$ reside in a very high-dimensional space, but only a relatively small number of the components of $\boldsymbol{g}_t^f$ are non-zero. Such settings are prevalent, for instance, in text-based applications: in text categorization, the full dimension corresponds to the dictionary or set of tokens that is being employed while each gradient is typically computed from a single or a few documents, each of which contains words and bigrams constituting only a small subset of the full dictionary. The need to cope with gradient sparsity becomes further pronounced in mixed-norm problems, as a single component of the gradient may correspond to an entire row of $W$. Updating the entire matrix because a few entries of $\boldsymbol{g}_t^f$ are non-zero is clearly undesirable. Thus, we would like to extend our methods to cope efficiently with gradient sparsity. For concreteness, we focus in this section on the efficient implementation of $\ell_1$, $\ell_2$, and $\ell_\infty$ regularization, since the extension to mixed-norms (as in the previous section) is straightforward. We postpone the proof of the following proposition to Appendix C.

**Proposition 4.** *Let $\boldsymbol{w}_T$ be the end result of solving a succession of $T$ self-similar optimization problems for $t = 1, \ldots, T$,*

$$\mathcal{P}.1: \quad \boldsymbol{w}_t = \underset{\boldsymbol{w}}{\operatorname{argmin}} \frac{1}{2}\|\boldsymbol{w} - \boldsymbol{w}_{t-1}\|^2 + \lambda_t \|\boldsymbol{w}\|_q \ . \tag{10}$$

*Let $\boldsymbol{w}^\star$ be the optimal solution of the following optimization problem,*

$$\mathcal{P}.2: \quad \boldsymbol{w}^\star = \underset{\boldsymbol{w}}{\operatorname{argmin}} \frac{1}{2}\|\boldsymbol{w} - \boldsymbol{w}_0\|^2 + \left(\sum_{t=1}^{T} \lambda_t\right)\|\boldsymbol{w}\|_q \ . \tag{11}$$

*For $q \in \{1, 2, \infty\}$ the vectors $\boldsymbol{w}_T$ and $\boldsymbol{w}^\star$ are identical.*

The algorithmic consequence of Proposition 4 is that it is possible to perform a lazy update on each iteration by omitting the terms of $\boldsymbol{w}_t$ (or whole rows of the matrix $W_t$ when using mixed-norms) that are outside the support of $\boldsymbol{g}_t^f$, the gradient of the loss at iteration $t$. We do need to maintain the step-sizes used on each iteration and have them readily available on future rounds when we newly update coordinates of $\boldsymbol{w}$ or $W$. Let $\Lambda_t$ denote the sum of the step sizes times regularization multipliers $\lambda\eta_t$ used from round $1$ through $t$. Then a simple algebraic manipulation yields that instead of solving $\boldsymbol{w}_{t+1} = \operatorname{argmin}_{\boldsymbol{w}} \left\{ \frac{1}{2}\|\boldsymbol{w} - \boldsymbol{w}_t\|_2^2 + \lambda\eta_t\|\boldsymbol{w}\|_q \right\}$ repeatedly when $\boldsymbol{w}_t$ is not changing, we can simply cache the last time $t_0$ that $\boldsymbol{w}$ (or a coordinate in $\boldsymbol{w}$ or a row from $W$) was updated and, when it is needed, solve $\boldsymbol{w}_{t+1} = \operatorname{argmin}_{\boldsymbol{w}} \left\{ \frac{1}{2}\|\boldsymbol{w} - \boldsymbol{w}_t\|_2^2 + (\Lambda_t - \Lambda_{t_0})\|\boldsymbol{w}\|_q \right\}$. The advantage of the lazy evaluation is pronounced when using mixed-norm regularization as it lets us avoid updating entire rows so long as the row index corresponds to a zero entry of the gradient $\boldsymbol{g}_t^f$. In sum, at the expense of keeping a time stamp $t$ for each entry of $\boldsymbol{w}$ or row of $W$ and maintaining the cumulative sums $\Lambda_1, \Lambda_2, \ldots$, we get $O(k)$ updates of $\boldsymbol{w}$ when the gradient $\boldsymbol{g}_t^f$ has only $k$ non-zero components.

# 6 Experiments

In this section we compare FOBOS to state-of-the-art optimizers to demonstrate its relative merits and weaknesses. We perform more substantial experiments in the full version of the paper [13].

$\ell_2^2$ **and $\ell_1$-regularized experiments:** We performed experiments using FOBOS to solve both $\ell_1$ and $\ell_2$-regularized learning problems. For the $\ell_2$-regularized experiments, we compared FOBOS to Pegasos [14], a fast projected gradient solver for SVM. Pegasos was originally implemented and evaluated on SVM-like problems by using the the hinge-loss as the empirical loss function along with an $\ell_2^2$ regularization term, but it can be straightforwardly extended to the binary logistic loss function. We thus experimented with both

$$f(\boldsymbol{w}) = \sum_{i=1}^{m} [1 - y_i \langle \boldsymbol{x}_i, \boldsymbol{w} \rangle]_+ \ \text{(hinge)} \quad \text{and} \quad f(\boldsymbol{w}) = \sum_{i=1}^{m} \log\left(1 + e^{-y_i \langle \boldsymbol{x}_i, \boldsymbol{w} \rangle}\right) \ \text{(logistic)}$$

as loss functions. To generate data for our experiments, we chose a vector $\boldsymbol{w}$ with entries distributed normally with $\boldsymbol{0}$ mean and unit variance, while randomly zeroing 50% of the entries in the vector.

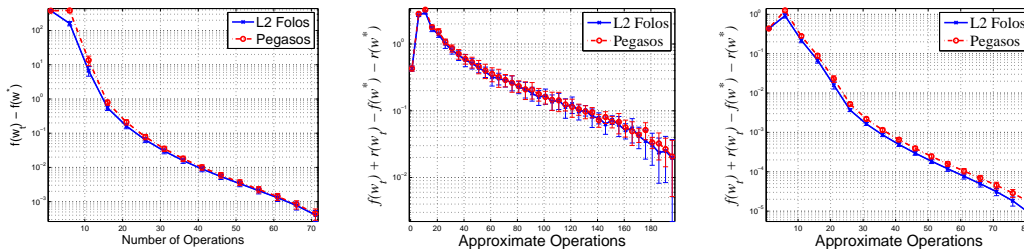

Figure 1: Comparison of FOBOS with Pegasos on the problems of logistic regression (left and right) and SVM (middle). The rightmost plot shows the performance of the algorithms without projection.

The examples $\boldsymbol{x}_i \in \mathbb{R}^n$ were also chosen at random with entries normally distributed. To generate target values, we set $y_i = \text{sign}(\langle \boldsymbol{x}_i, \boldsymbol{w} \rangle)$, and flipped the sign of 10% of the examples to add label noise. In all experiments, we used 1000 training examples of dimension 400.

The graphs of Fig. 1 show (on a log-scale) the regularized empirical loss of the algorithms minus the optimal value of the objective function. These results were averaged over 20 independent runs of the algorithms. In all experiments with the regularizer $\frac{1}{2}\lambda \|\boldsymbol{w}\|_2^2$, we used step size $\eta_t = \lambda/t$ to achieve logarithmic regret. The two left graphs of Fig. 1 show that FOBOS performs comparably to Pegasos on the logistic loss (left figure) and hinge (SVM) loss (middle figure). Both algorithms quickly approach the optimal value. In these experiments we let both Pegasos and FOBOS employ a projection after each gradient step into a 2-norm ball containing $\boldsymbol{w}^\star$ (see [14]). However, in the experiment corresponding to the rightmost plot of Fig. 1, we eliminated this additional projection step and ran the algorithms with the logistic loss. In this case, FOBOS slightly outperforms Pegasos. We hypothesize that the slightly faster rate of FOBOS is due to the explicit *shrinkage* that FOBOS performs in the $\ell_2$ update (see Eq. (7)).

In the next experiment, whose results are given in Fig. 2, we solved $\ell_1$-regularized logistic regression problems. We compared FOBOS to a simple subgradient method, where the subgradient of the $\lambda \|\boldsymbol{w}\|_1$ term is simply $\lambda \text{sign}(\boldsymbol{w})$), and a fast interior point (IP) method which was designed specifically for solving $\ell_1$-regularized logistic regression [15]. On the left side of Fig. 2 we show the objective function (empirical loss plus the $\ell_1$ regularization term) obtained by each of the algorithms minus the optimal objective value. We again used 1000 training examples of dimension 400. The learning rate was set to $\eta_t \propto 1/\sqrt{t}$. The standard subgradient method is clearly much slower than the other two methods even though we chose the initial step size for which the subgradient method converged the fastest. Furthermore, the subgradient method does *not* achieve any sparsity along its entire run. FOBOS quickly gets close to the optimal value of the objective function, but eventually the specialized IP method's asymptotically faster convergence causes it to surpass FOBOS. In order to obtain a weight vector $\boldsymbol{w}_t$ such that $f(\boldsymbol{w}_t) - f(\boldsymbol{w}^\star) \leq 10^{-2}$, FOBOS works very well, though the IP method enjoys faster convergence rate when the weight vector is very close to optimal solution. However, the IP algorithm was specifically designed to minimize empirical logistic loss with $\ell_1$ regularization whereas FOBOS enjoys a broad range of applicable settings.

The middle plot in Fig. 2 shows the sparsity levels (fraction of non-zero weights) achieved by FOBOS as a function of the number of iterations of the algorithm. Each line represents a different synthetic experiment as $\lambda$ is modified to give more or less sparsity to the solution vector $\boldsymbol{w}^\star$. The results show that FOBOS quickly selects the sparsity pattern of $\boldsymbol{w}^\star$, and the level of sparsity persists throughout its execution. We found this sparsity pattern common to non-stochastic versions of FOBOS we tested.

**Mixed-norm experiments:** Our experiments with mixed-norm regularization ($\ell_1/\ell_2$ and $\ell_1/\ell_\infty$) focus mostly on sparsity rather than on the speed of minimizing the objective. Our restricted focus is a consequence of the relative paucity of benchmark methods for learning problems with mixed-norm regularization. Our methods, however, as described in Sec. 4, are quite simple to implement, and we believe could serve as benchmarks for other methods to solve mixed-norm problems.

Our experiments compared multiclass classification with $\ell_1$, $\ell_1/\ell_2$, and $\ell_1/\ell_\infty$ regularization on the MNIST handwritten digit database and the StatLog Landsat Satellite dataset [16]. The MNIST database consists of 60,000 training examples and a 10,000 example test set with 10 classes. Each digit is a $28 \times 28$ gray scale image represented as a 784 dimensional vector. Linear classifiers

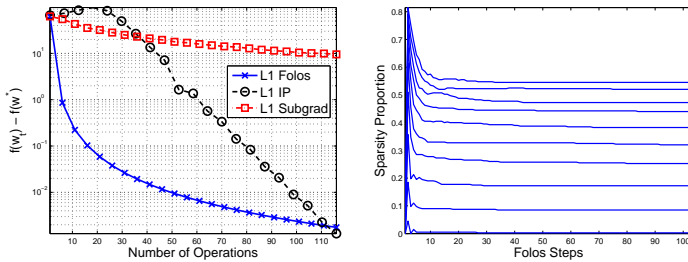

Figure 2: Left: Performance of FOBOS, a subgradient method, and an interior point method on $\ell_1$-regularized logistic regularization. Left: sparsity level achieved by FOBOS along its run.

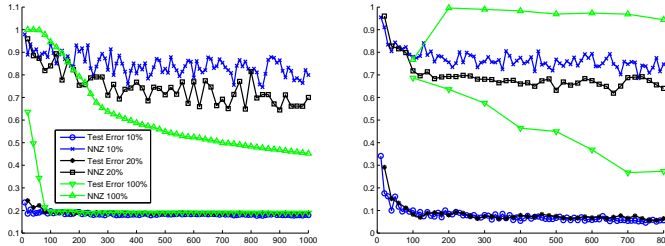

Figure 3: Left: FOBOS sparsity and test error for LandSat dataset with $\ell_1$-regularization. Right: FOBOS sparsity and test error for MNIST dataset with $\ell_1/\ell_2$-regularization.

do not perform well on MNIST. Thus, rather than learning weights for the original features, we learn the weights for classifier with Gaussian kernels, where value of the $j^{th}$ feature for the $i^{th}$ example is $x_{ij} = K(\boldsymbol{z}_i, \boldsymbol{z}_j) = e^{-\frac{1}{2}\|\boldsymbol{z}_i - \boldsymbol{z}_j\|^2}$. For the LandSat dataset we attempt to classify $3 \times 3$ neighborhoods of pixels in a satellite image as a particular type of ground, and we expanded the input 36 features into 1296 features by taking the product of all features.

In the left plot of Fig. 3, we show the test set error and row sparsity in $W$ as a function of training time (number of single-example gradient calculations) for the $\ell_1$-regularized multiclass logistic loss with 720 training examples. The green lines show results for using all 720 examples to calculate the gradient, black using 20% of the examples, and blue using 10% of the examples to perform stochastic gradient. Each used the same learning rate $\eta_t$, and the reported results are averaged over 5 independent runs with different training data. The righthand figure shows a similar plot but for MNIST with 10000 training examples and $\ell_1/\ell_2$-regularization. The objective value in training has a similar contour to the test loss. It is interesting to note that very quickly, FOBOS with stochastic gradient descent gets to its minimum test classification error, and as the training set size increases this behavior is consistent. However, the deterministic version increases the level of sparsity throughout its run, while the stochastic-gradient version has highly variable sparsity levels and does not give solutions as sparse as the deterministic counterpart. The slowness of non-stochastic gradient mitigates this effect for the larger sample size on MNIST in the right figure, but for longer training times, we do indeed see similar behavior.

For comparison of the different regularization approaches, we report in Table 1 the test error as a function of row sparsity of the learned matrix $W$. For the LandSat data, we see that using the block $\ell_1/\ell_2$ regularizer yields better performance for a given level of structural sparsity. However, on the MNIST data the $\ell_1$ regularization and the $\ell_1/\ell_2$ achieve comparable performance for each level of structural sparsity. Moreover, for a given level of structural sparsity, the $\ell_1$-regularized solution matrix $W$ attains significantly higher overall sparsity, roughly 90% of the entries of each non-zero row are zero. The performance on the different datasets might indicate that structural sparsity is effective only when the set of parameters indeed exhibit natural grouping.

| % Non-zero | $\ell_1$ Test | $\ell_1/\ell_2$ Test | $\ell_1/\ell_\infty$ Test | $\ell_1$ Test | $\ell_1/\ell_2$ Test | $\ell_1/\ell_\infty$ Test |
|---|---|---|---|---|---|---|
| 5 | .43 | **.29** | .40 | .37 | **.36** | .47 |
| 10 | .30 | **.25** | .30 | **.26** | **.26** | .31 |
| 20 | .26 | **.22** | .26 | **.15** | **.15** | .24 |
| 40 | .22 | **.19** | .22 | **.08** | **.08** | .16 |

Table 1: LandSat (left) and MNIST (right) classification error versus sparsity

# References

[1] D.P. Bertsekas. *Nonlinear Programming*. Athena Scientific, 1999.

[2] P. Zhao and B. Yu. On model selection consistency of Lasso. *Journal of Machine Learning Research*, 7:2541–2567, 2006.

[3] N. Meinshausen and P. Bühlmann. High dimensional graphs and variable selection with the Lasso. *Annals of Statistics*, 34:1436–1462, 2006.

[4] S. Wright, R. Nowak, and M. Figueiredo. Sparse reconstruction by separable approximation. In *IEEE International Conference on Acoustics, Speech, and Signal Processing*, pages 3373–3376, 2008.

[5] P. Combettes and V. Wajs. Signal recovery by proximal forward-backward splitting. *Multiscale Modeling and Simulation*, 4(4):1168–1200, 2005.

[6] J. Duchi, S. Shalev-Shwartz, Y. Singer, and T. Chandra. Efficient projections onto the $\ell_1$-ball for learning in high dimensions. In *Proceedings of the 25th International Conference on Machine Learning*, 2008.

[7] M. Zinkevich. Online convex programming and generalized infinitesimal gradient ascent. In *Proceedings of the Twentieth International Conference on Machine Learning*, 2003.

[8] E. Hazan, A. Kalai, S. Kale, and A. Agarwal. Logarithmic regret algorithms for online convex optimization. In *Proceedings of the Nineteenth Annual Conference on Computational Learning Theory*, 2006.

[9] G. Obozinski, M. Wainwright, and M. Jordan. High-dimensional union support recovery in multivariate regression. In *Advances in Neural Information Processing Systems 22*, 2008.

[10] J. Langford, L. Li, and T. Zhang. Sparse online learning via truncated gradient. In *Advances in Neural Information Processing Systems 22*, 2008.

[11] S. Shalev-Shwartz and A. Tewari. Stochastic methods for $\ell_1$-regularized loss minimization. In *Proceedings of the 26th International Conference on Machine Learning*, 2009.

[12] Y. Nesterov. *Introductory Lectures on Convex Optimization*. Kluwer Academic Publishers, 2004.

[13] J. Duchi and Y. Singer. Efficient online and batch learning using forward-backward splitting. *Journal of Machine Learning Research*, 10:In Press, 2009.

[14] S. Shalev-Shwartz, Y. Singer, and N. Srebro. Pegasos: Primal estimated sub-gradient solver for SVM. In *Proceedings of the 24th International Conference on Machine Learning*, 2007.

[15] K. Koh, S.J. Kim, and S. Boyd. An interior-point method for large-scale $\ell_1$-regularized logistic regression. *Journal of Machine Learning Research*, 8:1519–1555, 2007.

[16] D. Spiegelhalter and C. Taylor. *Machine Learning, Neural and Statistical Classification*. Ellis Horwood, 1994.

[17] R.T. Rockafellar. *Convex Analysis*. Princeton University Press, 1970.

